# A Practical Monte Carlo Implementation of Bayesian Learning

**Carl Edward Rasmussen**
Department of Computer Science
University of Toronto
Toronto, Ontario, M5S 1A4, Canada
carl@cs.toronto.edu

## Abstract

A practical method for Bayesian training of feed-forward neural networks using sophisticated Monte Carlo methods is presented and evaluated. In reasonably small amounts of computer time this approach outperforms other state-of-the-art methods on 5 data-limited tasks from real world domains.

## 1 INTRODUCTION

Bayesian learning uses a prior on model parameters, combines this with information from a training set, and then integrates over the resulting posterior to make predictions. With this approach, we can use large networks without fear of overfitting, allowing us to capture more structure in the data, thus improving prediction accuracy and eliminating the tedious search (often performed using cross validation) for the model complexity that optimises the bias/variance tradeoff. In this approach the size of the model is limited only by computational considerations.

The application of Bayesian learning to neural networks has been pioneered by MacKay (1992), who uses a Gaussian approximation to the posterior weight distribution. However, the Gaussian approximation is poor because of multiple modes in the posterior. Even locally around a mode the accuracy of the Gaussian approximation is questionable, especially when the model is large compared to the amount of training data.

Here I present and test a Monte Carlo method (Neal, 1995) which avoids the Gaussian approximation. The implementation is complicated, but the user is not required to have extensive knowledge about the algorithm. Thus, the implementation represents a practical tool for learning in neural nets.

## 1.1 THE PREDICTION TASK

The training data consists of $n$ examples in the form of inputs $\mathbf{x} = \{x^{(i)}\}$ and corresponding outputs $\mathbf{y} = \{y^{(i)}\}$ where $i = 1 \ldots n$. For simplicity we consider only real-valued scalar outputs. The network is parametrised by weights $\mathbf{w}$, and hyperparameters $\mathbf{h}$ that control the distributions for weights, playing a role similar to that of conventional weight decay. Weights and hyperparameters are collectively termed $\theta$, and the network function is written as $F_\theta(x)$, although the function value is only indirectly dependent on the hyperparameters (through the weights).

Bayes' rule gives the posterior distribution for the parameters in terms of the likelihood, $p(\mathbf{y}|\mathbf{x}, \theta)$, and prior, $p(\theta)$:

$$p(\theta|\mathbf{x}, \mathbf{y}) = \frac{p(\theta)p(\mathbf{y}|\mathbf{x}, \theta)}{p(\mathbf{y}|\mathbf{x})}.$$

To minimize the expected squared error on an unseen test case with input $\mathbf{x}^{(n+1)}$, we use the mean prediction

$$\hat{y}^{(n+1)} = \int F_\theta(x^{(n+1)})p(\theta|\mathbf{x}, \mathbf{y})d^k\theta. \tag{1}$$

## 2   MONTE CARLO SAMPLING

The following implementation is due to Neal (1995). The network weights are updated using the *hybrid Monte Carlo* method (Duane et al. 1987). This method combines the Metropolis algorithm with dynamical simulation. This helps to avoid the random walk behavior of simple forms of Metropolis, which is essential if we wish to explore weight space efficiently. The hyperparameters are updated using Gibbs sampling.

## 2.1   NETWORK SPECIFICATION

The networks used here are always of the same form: a single linear output unit, a single hidden layer of tanh units and a task dependent number of input units. All layers are fully connected in a feed forward manner (including direct connections from input to output). The output and hidden units have biases.

The network priors are specified in a hierarchical manner in terms of hyperparameters; weights of different kinds are divided into groups, each group having it's own prior. The output-bias is given a zero-mean Gaussian prior with a std. dev. of $\sigma = 1000$, so it is effectively unconstrained.

The hidden-biases are given a two layer prior: the bias $b$ is given a zero-mean Gaussian prior $b \sim \mathcal{N}(0, \sigma^2)$; the value of $\sigma$ is specified in terms of *precision* $\tau = \sigma^{-2}$, which is given a Gamma prior with mean $\mu = 400$ (corresponding to $\sigma = 0.05$) and shape parameter $\alpha = 0.5$; the Gamma density is given by $p(\tau) \sim \text{Gamma}(\mu, \alpha) \propto \tau^{\alpha/2-1} \exp(-\tau\alpha/2\mu)$. Note that this type of prior introduces a dependency between the biases for different hidden units through the common $\tau$. The prior for the hidden-to-output weights is identical to the prior for the hidden-biases, except that the variance of these weights under the prior is scaled down by the square root of the number of hidden units, such that the network output magnitude becomes independent of the number of hidden units. The noise variance is also given a Gamma prior with these parameters.

The input-to-hidden weights are given a three layer prior: again each weight is given a zero-mean Gaussian prior $w \sim \mathcal{N}(0, \sigma^2)$; the corresponding precision for the weights out of input unit $i$ is given a Gamma prior with a mean $\mu$ and a shape parameter $\alpha_1 = 0.5$: $\tau_i \sim \mathrm{Gamma}(\mu, \alpha_1)$. The mean $\mu$ is determined on the top level by a Gamma distribution with mean and shape parameter $\alpha_0 = 1$: $\mu_i \sim \mathrm{Gamma}(400, \alpha_0)$. The direct input-to-output connections are also given this prior.

The above-mentioned 3 layer prior incorporates the idea of Automatic Relevance Determination (ARD), due to MacKay and Neal, and discussed in Neal (1995). The hyperparameters, $\tau_i$, associated with individual inputs can adapt according to the relevance of the input; for an unimportant input, $\tau_i$ can grow very large (governed by the top level prior), thus forcing $\sigma_i$ and the associated weights to vanish.

## 2.2   MONTE CARLO SPECIFICATION

Sampling from the posterior weight distribution is performed by iteratively updating the values of the network weights and hyperparameters. Each iteration involves two components: weight updates and hyperparameter updates. A cursory description of these steps follows.

### 2.2.1   Weight Updates

Weight updates are done using the hybrid Monte Carlo method. A fictitious dynamical system is generated by interpreting weights as positions, and augmenting the weights **w** with momentum variables **p**. The purpose of the dynamical system is to give the weights "inertia" so that slow random walk behaviour can be avoided during exploration of weight space. The total energy, $H$, of the system is the sum of the kinetic energy, $K$, (a function of the momenta) and the potential energy, $E$. The potential energy is defined such that $p(\mathbf{w}) \propto \exp(-E)$. We sample from the joint distribution for **w** and **p** given by $p(\mathbf{w}, \mathbf{p}) \propto \exp(-E - K)$, under which the marginal distribution for **w** is given by the posterior. A sample of weights from the posterior can therefore be obtained by simply ignoring the momenta.

Sampling from the joint distribution is achieved by two steps: 1) finding new points in phase space with near-identical energies $H$ by simulating the dynamical system using a discretised approximation to Hamiltonian dynamics, and 2) changing the energy $H$ by doing Gibbs sampling for the momentum variables.

**Hamiltonian Dynamics.** Hamilton's first order differential equations for $H$ are approximated by a series of discrete first order steps (specifically by the *leapfrog* method). The first derivatives of the network error function enter through the derivative of the potential energy, and are computed using backpropagation. In the original version of the hybrid Monte Carlo method the final position is then accepted or rejected depending on the final energy $H^*$ (which is not necessarily equal to the initial energy $H$ because of the discretisation). Here we use a modified version that uses an average over a window of states instead. The step size of the discrete dynamics should be as large as possible while keeping the rejection rate low. The step sizes are set individually using several heuristic approximations, and scaled by an overall parameter $\varepsilon$. We use $L = 200$ iterations, a window size of 20 and a step size of $\varepsilon = 0.2$ for all simulations.

**Gibbs Sampling for Momentum Variables.** The momentum variables are updated using a modified version of Gibbs sampling, allowing the energy $H$ to change. A "persistence" of 0.95 is used; the new value of the momentum is a weighted sum of the previous value (weight 0.95) and the value obtained by Gibbs sampling (weight $(1 - 0.95^2)^{1/2}$). With this form of persistence, the momenta

changes approx. 20 times more slowly, thus increasing the "inertia" of the weights, so as to further help in avoiding random walks. Larger values of the persistence will further increase the weight inertia, but reduce the rate of exploration of $H$. The advantage of increasing the weight inertia in this way rather than by increasing $L$ is that the hyperparameters are updated at shorter intervals, allowing them to adapt to the rapidly changing weights.

### 2.2.2   Hyperparameter Updates

The hyperparameters are updated using Gibbs sampling. The conditional distributions for the hyperparameters given the weights are of the Gamma form, for which efficient generators exist, except for the top-level hyperparameter in the case of the 3 layer priors used for the weights from the inputs; in this case the conditional distribution is more complicated and a form of rejection sampling is employed.

### 2.3   NETWORK TRAINING AND PREDICTION

The network training consists of two levels of initialisation before sampling for networks used for prediction. At the first level of initialisation the hyperparameters (variance of the Gaussians) are kept constant at 1, allowing the weights to grow during 1000 leapfrog iterations. Neglecting this phase can cause the network to get caught for a long time in a state where weights and hyperparameters are both very small.

The scheme described above is then invoked and run for as long as desired, eventually producing networks from the posterior distribution. The initial $1/3$ of these nets are discarded, since the algorithm may need time to reach regions of high posterior probability. Networks sampled during the remainder of the run are saved for making predictions.

The predictions are made using an average of the networks sampled from the posterior as an approximation to the integral in eq. (1). Since the output unit is linear the final prediction can be seen as coming from a huge (fully connected) ensemble net with appropriately scaled output weights. All the results reported here were for ensemble nets with 4000 hidden units. The size of the individual nets is given by the rule that we want at least as many network parameters as we have training examples (with a lower limit of 4 hidden units). We hope thereby to be well out of the underfitting region. Using even larger nets would probably not gain us much (in the face of the limited training data) and is avoided for computational reasons.

All runs used the parameter values given above. The only check that is necessary is that the rejection rate stays low, say below 5%; if not, the step size should be lowered. In all runs reported here, $\varepsilon = 0.2$ was adequate. The parameters concerning the Monte Carlo method and the network priors were all selected based on intuition and on experience with toy problems. Thus no parameters need to be set by the user.

## 3   TESTS

The performance of the algorithm was evaluated by comparing it to other state-of-the-art methods on 5 real-world regression tasks. All 5 data sets have previously been studied using a 10-way cross-validation scheme (Quinlan 1993). The tasks in these domains is to predict price or performance of an object from various discrete and real-valued attributes. For each domain the data is split into two sets of roughly equal size, one for training and one for testing. The training data is

further subdivided into full-, half-, quarter- and eighth-sized subsets, 15 subsets in total. Networks are trained on each of these partitions, and evaluated on the large common test set. On the small training sets, the average performance and one std. dev. error bars on this estimate are computed.

## 3.1 ALGORITHMS

The Monte Carlo method was compared to four other algorithms. For the three neural network methods nets with a single hidden layer and direct input-output connections were used. The Monte Carlo method was run for 1 hour on each of the small training sets, and 2, 4 and 8 hours respectively on the larger training sets. All simulations were done on a 200 MHz MIPS R4400 processor. The Gaussian Process method is described in a companion paper (Williams & Rasmussen 1996).

The Evidence method (MacKay 1992) was used for a network with separate hyper-parameters for the direct connections, the weights from individual inputs (ARD), hidden biases, and output biases. Nets were trained using a conjugate gradient method, allowing 10000 gradient evaluations (batch) before each of 6 updates of the hyperparameters. The network Hessian was computed analytically. The value of the evidence was computed without compensating for network symmetries, since this can lead to a vastly over-estimated evidence for big networks where the posterior Gaussians from different modes overlap. A large number of nets were trained for each task, with the number of hidden units computed from the results of previous nets by the following heuristics: The **min** and **max** number of hidden units in the 20% nets with the highest evidences were found. The new architecture is picked from a Gaussian (truncated at 0) with mean $(\mathbf{max} - \mathbf{min})/2$ and std. dev. $2 + \mathbf{max} - \mathbf{min}$, which is thought to give a reasonable trade-off between exploration and exploitation. This procedure is run for 1 hour of cpu time or until more than 1000 nets have been trained. The final predictions are made from an ensemble of the 20% (but a maximum of 100) nets with the highest evidence.

An ensemble method using cross-validation to search over a 2-dimensional grid for the number of hidden units and the value of a single weight decay parameter has been included, as an attempt to have a thorough version of "common practise". The weight decay parameter takes on the values 0, 0.01, 0.04, 0.16, 0.64 and 2.56. Up to 6 sizes of nets are used, from 0 hidden units (a linear model) up to a number that gives as many weights as training examples. Networks are trained with a conjugent gradient method for 10000 epochs on each of these up to 36 networks, and performance was monitored on a validation set containing 1/3 of the examples, selected at random. This was repeated 5 times with different random validation sets, and the architecture and weight decay that did best on average was selected. The predictions are made from an ensemble of 10 nets with this architecture, trained on the full training set. This algorithm took several hours of cpu time for the largest training sets.

The Multivariate Adaptive Regression Splines (MARS) method (Friedman 1991) was included as a non-neural network approach. It is possible to vary the maximum number of variables allowed to interact in the additive components of the model. It is common to allow either pairwise or full interactions. I do not have sufficient experience with MARS to make this choice. Therefore, I tried both options and reported for each partition on each domain the best performance based on the test error, so results as good as the ones reported here might not be obtainable in practise. All other parameters of MARS were left at their default values. MARS always required less than 1 minute of cpu time.

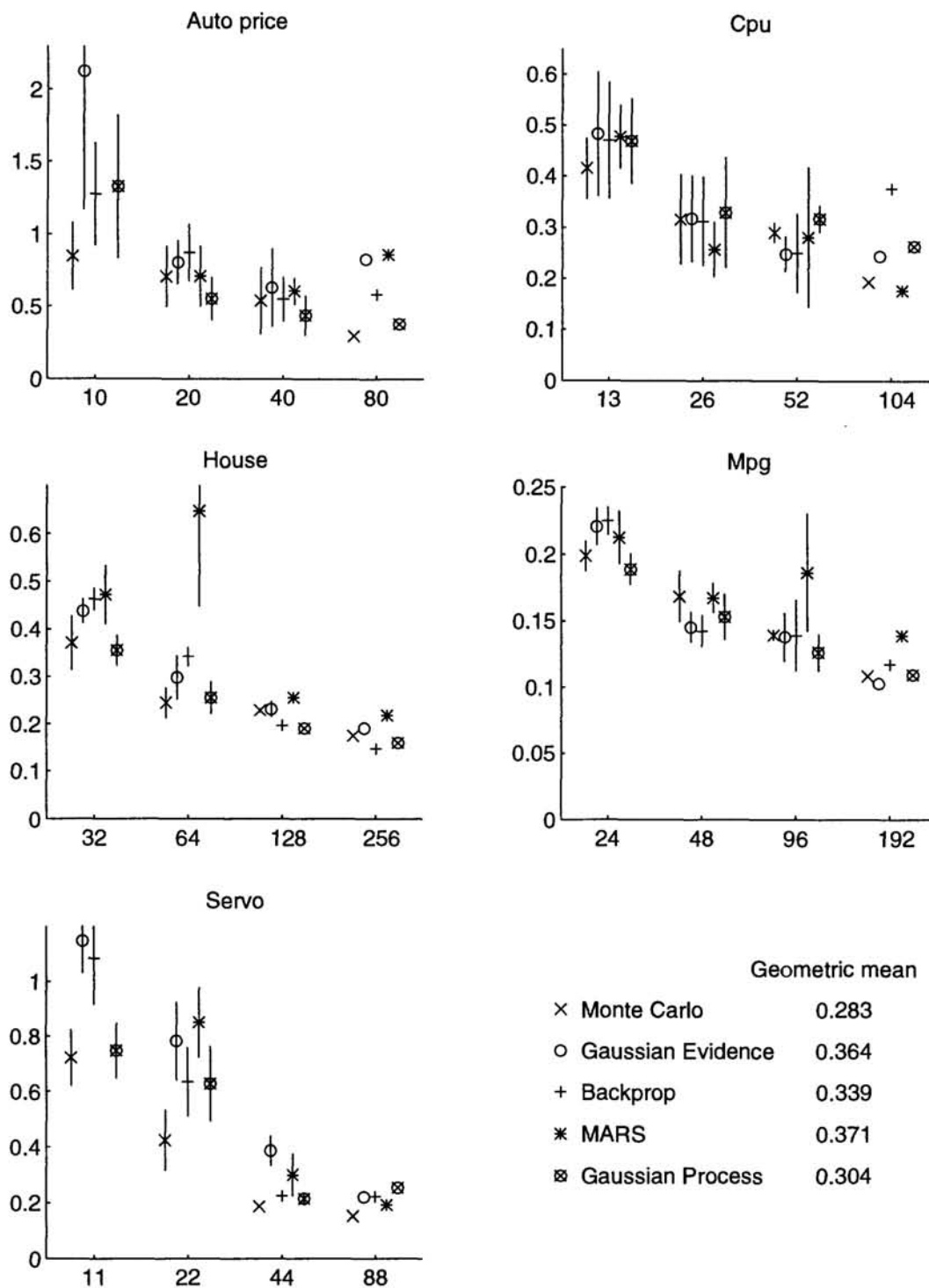

Figure 1: Squared error on test cases for the five algorithms applied to the five problems. Errors are normalized with respect to the variance on the test cases. The x-axis gives the number of training examples; four different set sizes were used on each domain. The error bars give one std. dev. for the distribution of the *mean* over training sets. No error bar is given for the largest size, for which only a single training set was available. Some of the large error bars are cut of at the top. MARS was unable to run on the smallest partitions from the **Auto price** and the **servo** domains; in these cases the means of the four other methods were used in the reported geometric mean for MARS.

Table 1: Data Sets

| domain | # training cases | # test cases | # binary inputs | # real inputs |
|--------|------------------|--------------|-----------------|---------------|
| Auto Price | 80 | 79 | 0 | 16 |
| Cpu | 104 | 105 | 0 | 6 |
| House | 256 | 250 | 1 | 12 |
| Mpg | 192 | 200 | 6 | 3 |
| Servo | 88 | 79 | 10 | 2 |

## 3.2 PERFORMANCE

The test results are presented in fig. 1. On the **servo** domain the Monte Carlo method is uniformly better than all other methods, although the difference should probably not always be considered statistically significant. The Monte Carlo method generally does well for the smallest training sets. Note that no single method does well on all these tasks. The Monte Carlo method is never vastly out-performed by the other methods.

The geometric mean of the performances over all 5 domains for the the 4 different training set sizes is computed. Assuming a Gaussian distribution of prediction errors, the log of the error variance can (apart from normalising constants) be interpreted as the amount of information unexplained by the models. Thus, the log of the geometric means in fig. 1 give the average information unexplained by the models. According to this measure the Monte Carlo method does best, closely followed by the Gaussian Process method. Note that MARS is the worst, even though the decision between pairwise and full interactions were made on the basis of the test errors.

## 4   CONCLUSIONS

I have outlined a black-box Monte Carlo implementation of Bayesian learning in neural networks, and shown that it has an excellent performance. These results suggest that Monte Carlo based Bayesian methods are serious competitors for practical prediction tasks on data limited domains.

### Acknowledgements

I am grateful to Radford Neal for his generosity with insight and software. This research was funded by a grant to G. Hinton from the Institute for Robotics and Intelligent Systems.

### References

S. Duane, A. D. Kennedy, B. J. Pendleton & D. Roweth (1987) "Hybrid Monte Carlo", *Physics Letters B*, vol. 195, pp. 216–222.

J. H. Friedman (1991) "Multivariate adaptive regression splines" (with discussion), *Annals of Statistics*, 19, 1-141 (March). Source: http://lib.stat.cmu.edu/general/mars3.5.

D. J. C. MacKay (1992) "A practical Bayesian framework for backpropagation networks", *Neural Computation*, vol. 4, pp. 448–472.

R. M. Neal (1995) *Bayesian Learning for Neural Networks*, PhD thesis, Dept. of Computer Science, University of Toronto, ftp: pub/radford/thesis.ps.Z from ftp.cs.toronto.edu.

J. R. Quinlan (1993) "Combining instance-based and model-based learning", *Proc. ML'93* (ed P.E. Utgoff), San Mateo: Morgan Kaufmann.

C. K. I. Williams & C. E. Rasmussen (1996). "Regression with Gaussian processes", *NIPS 8*, editors D. Touretzky, M. Mozer and M. Hesselmo. (this volume).